# A Nonparametric Approach to Bottom-Up Visual Saliency

**Wolf Kienzle, Felix A. Wichmann, Bernhard Schölkopf, and Matthias O. Franz**
Max Planck Institute for Biological Cybernetics,
Spemannstr. 38, 72076 Tübingen, Germany
{kienzle,felix,bs,mof}@tuebingen.mpg.de

## Abstract

This paper addresses the bottom-up influence of local image information on human eye movements. Most existing computational models use a set of biologically plausible linear filters, e.g., Gabor or Difference-of-Gaussians filters as a front-end, the outputs of which are nonlinearly combined into a real number that indicates visual saliency. Unfortunately, this requires many design parameters such as the number, type, and size of the front-end filters, as well as the choice of nonlinearities, weighting and normalization schemes etc., for which biological plausibility cannot always be justified. As a result, these parameters have to be chosen in a more or less ad hoc way. Here, we propose to *learn* a visual saliency model directly from human eye movement data. The model is rather simplistic and essentially parameter-free, and therefore contrasts recent developments in the field that usually aim at higher prediction rates at the cost of additional parameters and increasing model complexity. Experimental results show that—despite the lack of any biological prior knowledge—our model performs comparably to existing approaches, and in fact learns image features that resemble findings from several previous studies. In particular, its maximally excitatory stimuli have center-surround structure, similar to receptive fields in the early human visual system.

## 1 Introduction

The human visual system samples images through saccadic eye movements, which rapidly change the point of fixation. It is believed that the underlying mechanism is driven by both *top-down* strategies, such as the observer's task, thoughts, or intentions, and by *bottom-up* effects. The latter are usually attributed to early vision, *i.e.*, to a system that responds to simple, and often local image features, such as a bright spot in a dark scene. During the past decade, several studies have explored which image features attract eye movements. For example, *Reinagel* and *Zador* [18] found that contrast was substantially higher at gaze positions, *Krieger* et al. [10] reported differences in the intensity bispectra. *Parkhurst*, *Law*, and *Niebur* [13] showed that a *saliency map* [9], computed by a model similar to the widely used framework by *Itti*, *Koch* and *Niebur* [3, 4], is significantly correlated with human fixation patterns. Numerous other hypotheses were tested [1, 5, 6, 10, 12, 14, 16, 17, 19, 21], including intensity, edge content, orientation, symmetry, and entropy.

Each of the above models is built on a particular choice of image features that are believed to be relevant to visual saliency. A common approach is to compute several feature maps from linear filters that are biologically plausible, *e.g.*, Difference of Gaussians (DoG) or Gabor filters, and nonlinearly combine the feature maps into a single saliency map [1, 3, 4, 13, 16, 21]. This makes it straightforward to construct complex models from simple, biologically plausible components. A downside of this parametric approach, however, is that the feature maps are chosen manually by the designer. As a consequence, any such model is biased to certain image structure, and therefore discriminates features that might not seem plausible at first sight, but may well play a significant role.

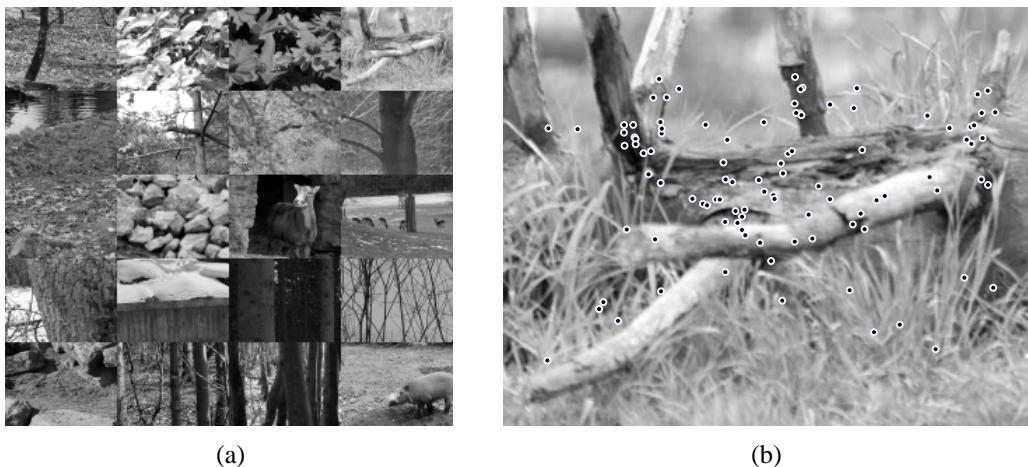

<div align="center">(a)                     (b)</div>

Figure 1: Eye movement data. (a) shows 20 (out of 200) of the natural scenes that were presented to the 14 subjects. (b) shows the top right image from (a), together with the recorded fixation locations from all 14 subjects. The average viewing time per subject was approximately 3 seconds.

Another problem comes from the large number of additional design parameters that are necessary in any implementation, such as the precise filter shapes, sizes, weights, nonlinearities, *etc*. While choices for these parameters are often only vaguely justified in terms of their biological plausibility, they greatly affect the behavior of the system as a whole and thus its predictive power. The latter, however, is often used as a measure of plausibility. This is clearly an undesirable situation, since it makes a fair comparison between models very difficult. In fact, we believe that this may explain the conflicting results in the debate about whether edges or contrast filters are more relevant [1, 6, 13].

In this paper we present a nonparametric approach to bottom-up saliency, which does not (or to a far lesser extent) suffer from the shortcomings described above. Instead of using a predefined set of feature maps, our saliency model is *learned* directly from human eye movement data. The model consists of a nonlinear mapping from an image patch to a real value, trained to yield positive outputs on fixated, and negative outputs on randomly selected image patches. The main difference to previous models is that our saliency function is essentially determined by the fact that it maximizes the prediction performance on the observed data. Below, we show that the prediction performance of our model is comparable to that of biologically motivated models. Furthermore, we analyze the system in terms of the features it has learned, and compare our findings to previous results.

## 2 Eye Movement Data

Eye movement data were taken from [8]. They consist of 200 natural images (1024×768, 8bit grayscale) and 18,065 fixation locations recorded from 14 naïve subjects. The subjects freely viewed each image for about three seconds on a 19 inch CRT at full screen size and 60cm distance, which corresponds to $37° \times 27°$ of visual angle. For more details about the recording setup, please refer to [8]. Figure 1 illustrates the data set.[1]

Below, we are going to formulate saliency learning as a classification problem. This requires negative examples, *i.e.*, a set of non-fixated, or background locations. As pointed out in [18, 21], care must be taken that no spurious differences in the local image statistics are generated by using different spatial distributions for positive and negative examples. As an example, fixation locations are usually biased towards the center of the image, probably due to the reduced physical effort when looking straight. At the same time, it is known that local image statistics can be correlated with

image location [18, 21], *e.g.*, due to the photographer's bias of keeping objects at the center of the image. If we sampled background locations uniformly over the image, our system might learn the difference between pixel statistics at the image center and towards the boundary, instead of the desired difference between fixated and non-fixated locations. Moreover, the learning algorithm might be mislead by simple boundary effects. To avoid this effect, we use the 18,065 fixation locations to generate an equal number of background locations by using the same image coordinates, but with the corresponding image numbers shuffled. This ensures that the spatial distributions of both classes are identical.

The proposed model computes saliency based on local image structure. To represent fixations and background locations accordingly, we cut out a square image patch at each location and stored the pixel values in a feature vector $\mathbf{x}_i$ together with a label $y_i \in \{1; -1\}$, indicating fixation or background. Unfortunately, choosing an appropriate patch size and resolution is not straightforward, as there might be a wide range of reasonable values. To remedy this, we follow the approach proposed in [8], which is a simple compromise between computational tractability and generality: we fix the resolution to $13 \times 13$ pixels, but leave the patch size $d$ unspecified, *i.e.*, we construct a separate data set for various values of $d$. Later, we determine the size $d$ which leads to the best generalization performance estimate. For each image location, 11 patches were extracted, with sizes ranging between $d = 0.47°$ and $d = 27°$ visual angle, equally spaced on a logarithmic scale. Each patch was subsampled to $13 \times 13$ pixels, after low-pass filtering to reduce aliasing effects. The range of sizes was chosen such that pixels in the smallest patch correspond to image pixels at full screen resolution, and that the largest patch has full screen height. Finally, for each patch we subtracted the mean intensity, and stored the normalized pixel values in a 169-dimensional feature vector $\mathbf{x}_i$.

The data were divided into a training (two thirds) and a test set (one third). This was done such that both sets contained data from all 200 images, but never from the same subject on the same image. For model selection (Section 4.1) and assessment (Section 4.2), which rely on cross-validation estimates of the generalization error, further splits were required. These splits were done image-wise, *i.e.*, such that no validation or test fold contained any data from images in the corresponding training fold. This is necessary, since image patches from different locations can overlap, leading to a severe over-estimation of the generalization performance.

## 3   Model and Learning Method

From the eye movement described in Section 2, we learn a bottom-up saliency map $f(\mathbf{x}) : \mathbb{R}^{169} \to \mathbb{R}$ using a support vector machine (SVM) [2]. We model saliency as a linear combination of Gaussian radial basis functions (RBFs), centered at the training points $\mathbf{x}_i$,

$$f(\mathbf{x}) = \sum_{i=1}^{m} \alpha_i y_i \exp\left(-\frac{\|\mathbf{x} - \mathbf{x}_i\|^2}{2\sigma^2}\right). \tag{1}$$

The SVM algorithm determines non-negative coefficients $\alpha_i$ such that the regularized risk $R(f) = D(f) + \lambda S(f)$ is minimized. Here, $D(f)$ denotes the data fit $\sum_{i=1}^{m} \max(0, \ 1 - y_i f(\mathbf{x}_i))$, and $S(f)$ is the standard SVM regularizer $\frac{1}{2}\|f\|^2$ [2]. The tradeoff between data fit and smoothness is controlled by the parameter $\lambda$. As described in Section 4.1, this design parameter, as well as the RBF bandwidth $\sigma$ and the patch size $d$ is determined by maximizing the model's estimated prediction performance.

It is insightful to compare our model (1) to existing models. Similar to most existing approaches, our model is based on linear filters whose outputs are nonlinearly combined into a real-valued saliency measure. This is a common model for the early visual system, and receptive-field estimation techniques such as reverse-correlation usually make the same assumptions. It differs from existing approaches in terms of its nonparametric nature, *i.e.*, the basic linear filters are the training samples themselves. That way, the system is not restricted to the designer's choice of feature maps, but learns relevant structure from the data. For the nonlinear component, we found the Gaussian RBF appropriate for two reasons: first, it is a universal SVM kernel [20], allowing the model to approximate any smooth function on the data points; second, it carries no information about the spatial ordering of the pixels within an image patch $\mathbf{x}$: if we consistently permuted the pixels of the training and test patches, the model output would be identical. This implies that the system is has no *a priori* preference for particular image structures. The SVM algorithm was chosen primarily since it is a

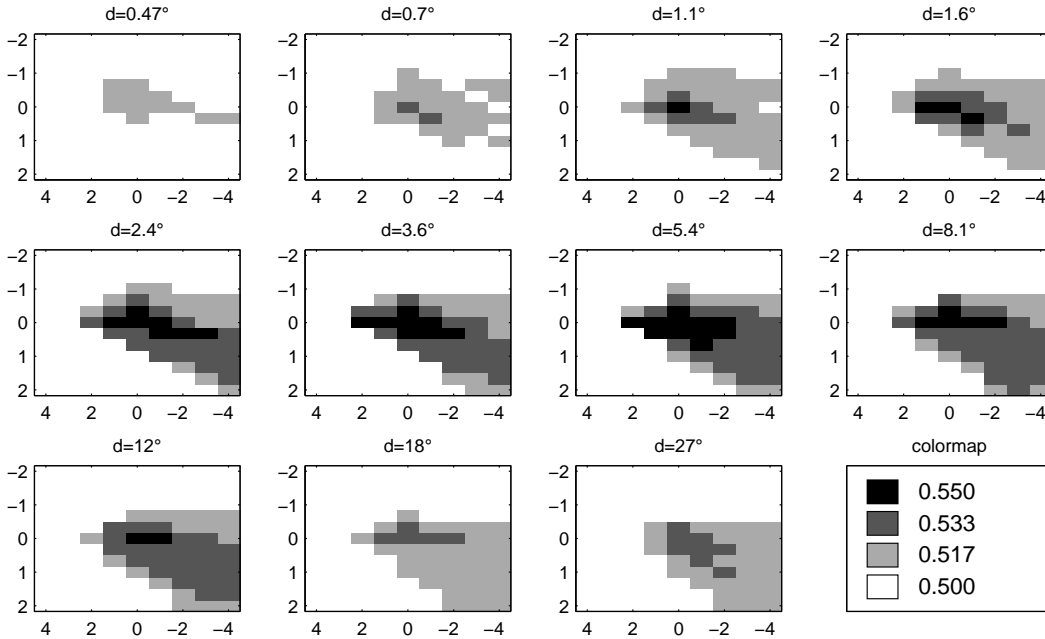

Figure 2: Selection of the parameters $d$, $\sigma$ and $\lambda$. Each panel shows the estimated model performance for a fixed $d$, and all $\sigma$ (vertical axes, label values denote $\log_{10} \sigma$) and $\lambda$ (horizontal axes, label values denote $\log_{10} \lambda$). Darker shades of gray denote higher accuracy; a legend is shown on the lower right. Based on these results, we fixed $d = 5.4°$, $\log_{10} \sigma = 0$, and $\log_{10} \lambda = 0$.

powerful standard method for binary classification. In light of its resemblance to regularized logistic regression, our method is therefore related to the one proposed in [1]. Their model is parametric, however.

# 4 Experiments

## 4.1 Selection of $d, \sigma$, and $\lambda$

For fixing $d, \sigma$, and $\lambda$, we conducted an exhaustive search on a $11 \times 9 \times 13$ grid with the grid points equally spaced on a log scale such that $d = 0.47°, \ldots, 27°$, $\sigma = 0.01, \ldots, 100$, and $\lambda = 0.001, \ldots, 10,000$. In order to make the search computationally tractable, we divided the training set (Section 2) into eight parts. Within each part, and for each point on the parameter grid, we computed a cross-validation estimate of the classification accuracy (*i.e.*, the relative frequency of $\text{sign} f(\mathbf{x}_i) = y_i$). The eight estimates were then averaged to yield one performance estimate for each grid point. Figure 2 illustrates the results. Each panel shows the model performance for one $(\sigma, \lambda)$-slice of the parameter space. The performance peaks at $0.55$ ($0.013$ standard error of mean, SEM) at $d = 5.4°, \sigma = 1, \lambda = 1$, which is in agreement with [8], up to their slightly different $d = 6.2°$.[2] Note that while $0.55$ is not much, it is four standard errors above chance level. Furthermore, all $(\sigma, \lambda)$ plots show the same, smooth pattern which is known to be characteristic for RBF-SVM model selection [7]. This further suggests that, despite the low absolute performance, our choice of parameters is well justified. Model performance (Section 4.2) and interpretation (Section 4.3) were qualitatively stable within at least one step in any direction of the parameter grid.

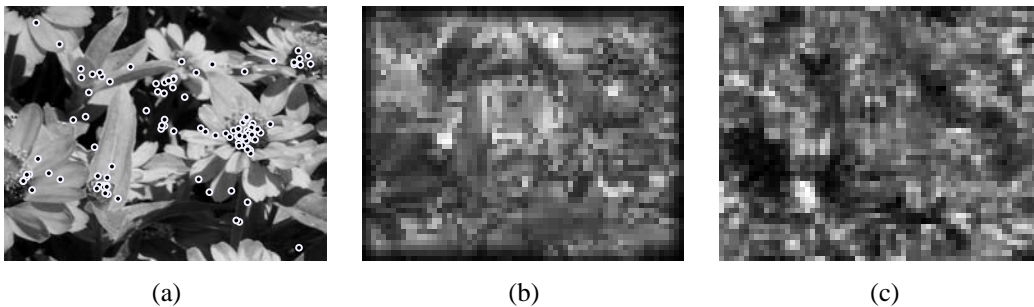

| (a) | (b) | (c) |

Figure 3: Saliency maps. (a) shows a natural scene from our database, together with the recorded eye movements from all 14 subjects. *Itti*'s saliency map, using "standard" normalization is shown in (b). Brighter regions denote more salient areas. The picture in (c) shows our learned saliency map, which was re-built for this example, with the image in (a) excluded from the training data. Note that the differing boundary effects are of no concern for our performance measurements, since hardly any fixations are that close to the boundary.

## 4.2 Model Performance

To test the model's performance with the optimal parameters ($d = 5.4°$, $\sigma = 1$, $\lambda = 1$) and more training examples, we divided the test set into eight folds. Again, this was done image-wise, *i.e.*, such that each fold comprised the data from 25 images (cf. Section 2). For each fold we trained our model on all training data *not* coming from the respective 25 images. As expected, the use of more training data significantly improved the accuracy to 0.60 (0.011 SEM). For a comparison with other work, we also computed the mean ROC score of our system, 0.64 (0.010 SEM). This performance is lower than the 0.67 reported in [8]. However, their model explains only about 10% of the "simplest" fixations in the data. Another recent study yielded 0.63 [21], although on a different data set. *Itti*'s model [4] was tested in [15], who report ROC scores around 0.65 (taken from a graph, no actual numbers are given). Scores of up to 0.70 were achieved with an extended version, that uses more elaborate long-range interactions and eccentricity-dependent processing. We also ran *Itti*'s model on our test set, using the code from [22]. We tried both the "standard" [3] and "iterative" [4] normalization scheme. The best performing setting was the earlier "standard" method, which yielded 0.62 (0.022 SEM). The more recent iterative scheme did not improve on this result, also not when only the first, or first few fixations were considered. For a qualitative comparison, Figure 3 shows our learned saliency map and *Itti*'s model evaluated on a sample image.

It is important to mention that the purpose of the above comparison is *not* to show that our model makes better predictions than existing models — which would be a weak statement anyway since the data sets are different. The main insight here is that our nonparametric model performs at the same level as existing, biologically motivated models, which implement plausible, multi-scale front-end filters, carefully designed non-linearities, and even global effects.

## 4.3 Feature Analysis

In the previous section we have shown that our model generalizes to unseen data, *i.e.*, that it has learned regularities in the data that are relevant to the human fixation selection mechanism. This section addresses the question of what the learned regularities are, and how they are related to existing models. As mentioned in Section 1, characterizing a nonlinear model solely by the feature maps at its basis is insufficient. In fact, our SVM-based model is an example where this would be particularly wrong. An SVM assigns the smaller (down to zero) weights $\alpha_i$, the easier the respective training samples $\mathbf{x}_i$ can be classified. Describing $f$ by its support vectors $\{\mathbf{x}_i | \alpha_i > 0\}$ is therefore misleading, since they represent unusual examples, rather than prototypes. To avoid this, we instead characterize the learned function by means of *inputs* $\mathbf{x}$ that are particularly excitatory or inhibitory to the *entire* system. As a first test, we collected 20,000 image patches from random locations in natural scenes (not in the training set) and presented them to our system. The top and bottom 100 patches sorted by model output and a histogram over all 20,000 saliency values are shown in Figure 4 . Note that since our model is unbiased towards any particular image structure, the different

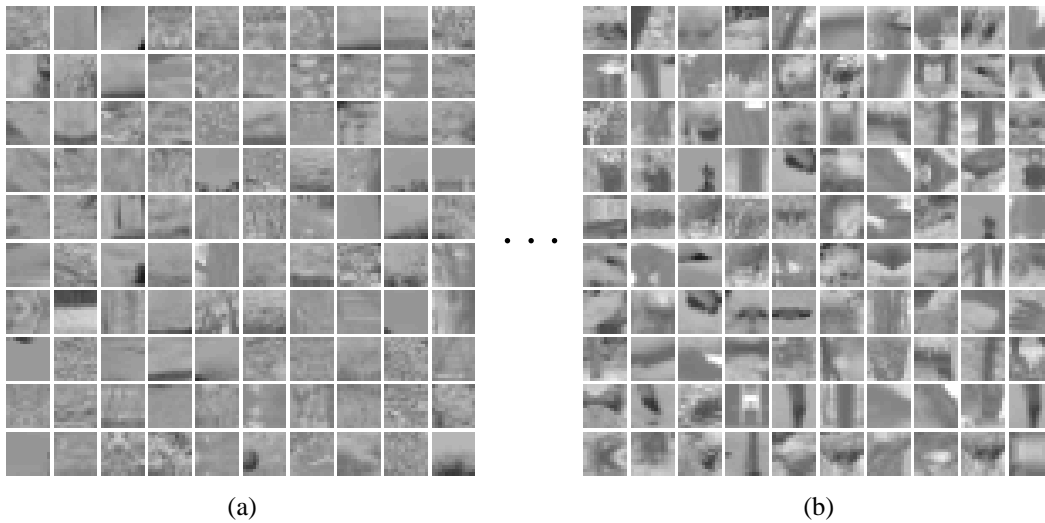

(a)                                                                (b)

Figure 4: Natural image patches ranked by saliency according to our model. The panels (a) and (b) show the bottom and top 100 of $20,000$ patches, respectively (the dots in between denote the $18,800$ patches which are not shown). A histogram of all $20,000$ saliency values is given on the lower right. The outputs in (a) range from $-2.0$ to $-1.7$, the ones in (b) from $0.99$ to $1.8$.

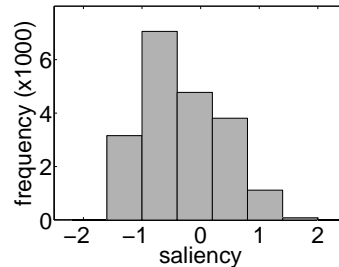

patterns observed in high and low output patches are solely due to differences between pixel statistics at fixated and background regions. The high output patches seem to have higher contrast, which is in agreement with previous results, *e.g.*, [8, 10, 14, 18]. In fact, the correlation coefficient of the model output (all $20,000$ values) with r.m.s. contrast is $0.69$. Another result from [14, 18] is that in natural images the correlation between pixel values decays faster at fixated locations, than at randomly chosen locations. Figure 4 shows this trend as well: as we move away from the patch center, the pixels' correlation with the center intensity decays faster for patches with high predicted salience. Moreover, a study on bispectra at fixated image locations [10] suggested that "the saccadic selection system avoids image regions, which are dominated by a single oriented structure. Instead, it selects regions containing different orientations, like occlusions, corners, *etc*". A closer look at Figure 4 reveals that our model tends to attribute saliency not alone to contrast, but also to non-trivial image structure. Extremely prominent examples of this effect are the high contrast edges appearing among the bottom 100 patches, *e.g.*, in the patches at position (7,2) or (10,10).

To further characterize the system, we explicitly computed the maximally excitatory and inhibitory stimuli. This amounts to solving the unconstrained optimization problems $\arg\max_{\mathbf{x}} f(\mathbf{x})$ and $\arg\min_{\mathbf{x}} f(\mathbf{x})$, respectively. Since $f$ is differentiable, we can use a simple gradient method. The only problem is that $f(\mathbf{x})$ can have multiple extrema in $\mathbf{x}$. A common way to deal with local optima is to run the search several times with different initial values for $\mathbf{x}$. Here, we repeated the search $1,000$ times for both minima and maxima. The initial $\mathbf{x}$ were constructed by drawing 169 pixel values from a normal distribution with zero mean and then normalizing the patch standard deviation to $0.11$ (the average value over the training patches). The $1,000$ optimal values were then clustered using $k$-means. The number of clusters $k$ was found by increasing $k$ until the clusters were stable. Interestingly, the clusters for both minima and maxima were already highly concentrated for $k = 2$, *i.e.*, within each cluster, the average variance of a pixel was less than $0.03\%$ of the pixel variance of its center patch. This result could also be confirmed visually, *i.e.*, despite the randomized initial values both optimization problems had only two visually distinct outcomes. We also re-ran this experiment with natural image patches as starting values, with identical results. This indicates that our saliency function has essentially two minima and two maxima in $\mathbf{x}$. The four optimal stimuli are shown in Figure 5 . The first two images (a) and (b) show the maximally inhibitory stimuli.

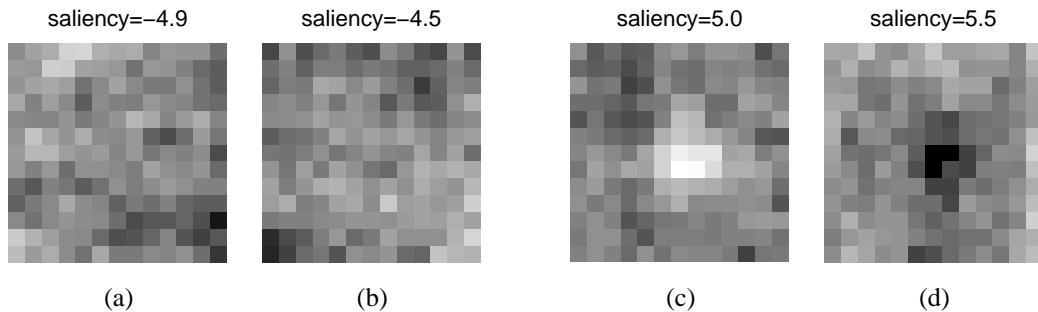

| saliency=−4.9 | saliency=−4.5 | saliency=5.0 | saliency=5.5 |
| :---: | :---: | :---: | :---: |
| (a) | (b) | (c) | (d) |

Figure 5: Maximally inhibitory and excitatory stimuli of the learned model. Note the large magnitude of the saliency values compared to the typical model output (cf. the histogram in Figure 4). (a) and (b): the two maximally inhibitory stimuli (lowest possible saliency). (c) and (d): the two maximally excitatory stimuli (highest possible saliency), (e) and (f): the radial average of (c) and (d), respectively.

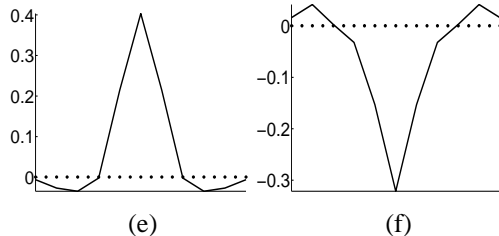

These are rather difficult to interpret other than no particular structure is visible. On the other hand, the maximally excitatory stimuli, denoted by (c) and (d), have center-surround structure. All four stimuli have zero mean, which is not surprising since during gradient search, both the initial value and the step directions—which are linear combinations of the training data—have zero mean. As a consequence, the surrounds of (c) and (d) are inhibitory w.r.t. their centers, which can also be seen from the different signs in their radial averages (e) and (f).[3] The optimal stimuli thus bear a close resemblance to receptive fields in the early visual system [11]. To see that the optimal stimuli have in fact prototype character, note how the histogram in Figure 4 reflects the typical distribution of natural image patches along the learned saliency function. It illustrates that the saliency values of unseen natural image patches usually lie between −2.0 and 1.8 (for the training data, they are between −1.8 and 2.2). In contrast, our optimal stimuli have saliency values of 5.0 and 5.5, indicating that they represent the difference between fixated and background locations in a much more articulated way than any of the noisy measurements in our data set.

## 5  Discussion

We have presented a nonparametric model for bottom-up visual saliency, trained on human eye movement data. A major goal of this work was to complement existing approaches in that we keep the number of assumptions low, and instead learn as much as possible from the data. In order to make this tractable, the model is rather simplistic, *e.g.*, it implements no long-range interactions within feature maps. Nevertheless, we found that the prediction performance of our system is comparable to that of parametric, biologically motivated models. Although no such information was used in the design of our model, we found that the learned features are consistent with earlier results on bottom-up saliency. For example, the outputs of our model are strongly correlated with local r.m.s. contrast [18]. Also, we found that the maximally excitatory stimuli of our system have center-surround structure, similar to DoG filters commonly used in early vision models [3, 13, 21]. This is a nontrivial result, since our model has no preference for any particular image features, i.e., a priori, *any* $13 \times 13$ image patch is equally likely to be an optimal stimulus. Recently, several authors have explored whether oriented (Gabor) or center-surround (DoG) features are more relevant to human eye movements. As outlined in Section 1, this is a difficult task: while some results indicate that both features perform equally well [21], others suggest that one [1] or the other [6, 13] are more relevant. Our results shed additional light on this discussion in favor of center-surround features.

## Footnotes

[1]In our initial study [8], these data were preprocessed further. In order to reduce the noise due to varying top-down effects, only those locations that are consistent among subjects were used. Unfortunately, while this leads to higher prediction scores, the resulting model is only valid for the reduced data set, which in that case is less than ten percent of the fixations. To better explain the entire data set, in the present work we instead retain all 18,065 fixations, *i.e.*, we trade performance for generality.

[2]Due to the subsampling (Section 2), the optimal patch size of $d = 5.4°$ leads to an effective saliency map resolution of $89 \times 66$ (the original image is $1024 \times 768$), which corresponds to $2.4$ pixels per visual degree. While this might seem low, note that similar resolutions have been suggested for bottom-up saliency: using *Itti*'s model with default parameters leads to a resolution of $64 \times 48$.

[3]Please note that the radial average curves in Figure 5 (e) and (f) do not necessarily sum to zero, since the patch area in (c) and (d) grows quadratically with its corresponding radius.

# References

[1] R. J. Baddeley and B. W. Tatler. High frequency edges (but not contrast) predict where we fixate: A bayesian system identification analysis. *Vision Research*, 46(18):2824–2833, 2006.

[2] C. J. C. Burges. A tutorial on support vector machines for pattern recognition. *Data Mining and Knowledge Discovery*, 2(2):121–167, 1998.

[5] L. Itti. Quantifying the contribution of low-level saliency to human eye movements in dynamic scenes. *Visual Cognition*, 12(6):1093–1123, 2005.

[6] L. Itti. Quantitative modeling of perceptual salience at human eye position. *Visual Cognition (in press)*, 2006.

[3] L. Itti, Koch C., and E. Niebur. A model of saliency-based visual attention for rapid scene analysis. *IEEE Transactions on Pattern Analysis and Machine Intelligence*, 20(11):1254–1259, 1998.

[4] L. Itti and C. Koch. A saliency-based search mechanism for overt and covert shifts of visual attention. *Vision Research*, 40(10-12):1489–1506, 2000.

[7] S. S. Keerthi and C. J. Lin. Asymptotic behaviors of support vector machines with gaussian kernel. *Neural Computation*, 15:1667–1689, 2003.

[8] W. Kienzle, F. A. Wichmann, B. Schölkopf, and M. O. Franz. Learning an interest operator from human eye movements. In *Beyond Patches Workshop, International Conference on Computer Vision and Pattern Recognition*, 2006.

[9] C. Koch and S. Ullman. Shifts in selective visual attention: towards the underlying neural circuitry. *Human Neurobiology*, 4(4):219–227, 1985.

[10] G. Krieger, I. Rentschler, G. Hauske, K. Schill, and C. Zetzsche. Object and scene analysis by saccadic eye-movements: an investigation with higher-order statistics. *Spatial Vision*, 3(2,3):201–214, 2000.

[11] S. W. Kuffler. Discharge patterns and functional organization of mammalian retina. *Journal of Neurophysiology*, 16(1):37–68, 1953.

[12] S. K. Mannan, K. H. Ruddock, and D. S. Wooding. The relationship between the locations of spatial features and those of fixations made during visual examination of briefly presented images. *Spatial Vision*, 10(3):165–88, 1996.

[13] D. J. Parkhurst, K. Law, and E. Niebur. Modeling the role of salience in the allocation of overt visual attention. *Vision Research*, 42(1):107–123, 2002.

[14] D. J. Parkhurst and E. Niebur. Scene content selected by active vision. *Spatial Vision*, 16(2):125–154, 2003.

[15] R. J. Peters, A. Iyer, C. Koch, and L. Itti. Components of bottom-up gaze allocation in natural scenes (poster). In *Vision Sciences Society (VSS) Annual Meeting*, 2005.

[16] C. M. Privitera and L. W. Stark. Algorithms for defining visual regions-of-interest: Comparison with eye fixations. *IEEE Transactions on Pattern Analysis and Machine Intelligence*, 22(9):970–982, 2000.

[17] R. Raj, W. S. Geisler, R. A. Frazor, and A. C. Bovik. Contrast statistics for foveated visual systems: Fixation selection by minimizing contrast entropy. *Journal of the Optical Society of America A.*, 22(10):2039–2049, 2005.

[18] P. Reinagel and A. M. Zador. Natural scene statistics at the center of gaze. *Network: Computation in Neural Systems*, 10(4):341–350, 1999.

[19] L. W. Renninger, J. Coughlan, P. Verghese, and J. Malik. An information maximization model of eye movements. In *Advances in Neural Information Processing Systems*, volume 17, pages 1121–1128, 2005.

[20] I. Steinwart. On the influence of the kernel on the consistency of support vector machines. *Journal of Machine Learning Research*, 2:67–93, 2001.

[22] D. Walther. *Interactions of visual attention and object recognition: computational modeling, algorithms, and psychophysics*. PhD thesis, California Institute of Technology, 2006.
